# Implicit encoding of prior probabilities in optimal neural populations

**Deep Ganguli and Eero P. Simoncelli**

Howard Hughes Medical Institute, and
Center for Neural Science
New York University
New York, NY 10003

{dganguli,eero}@cns.nyu.edu

Optimal coding provides a guiding principle for understanding the representation of sensory variables in neural populations. Here we consider the influence of a prior probability distribution over sensory variables on the optimal allocation of neurons and spikes in a population. We model the spikes of each cell as samples from an independent Poisson process with rate governed by an associated tuning curve. For this response model, we approximate the Fisher information in terms of the density and amplitude of the tuning curves, under the assumption that tuning width varies inversely with cell density. We consider a family of objective functions based on the expected value, over the sensory prior, of a functional of the Fisher information. This family includes lower bounds on mutual information and perceptual discriminability as special cases. In all cases, we find a closed form expression for the optimum, in which the density and gain of the cells in the population are power law functions of the stimulus prior. This also implies a power law relationship between the prior and perceptual discriminability. We show preliminary evidence that the theory successfully predicts the relationship between empirically measured stimulus priors, physiologically measured neural response properties (cell density, tuning widths, and firing rates), and psychophysically measured discrimination thresholds.

## 1   Introduction

Many bottom up theories of neural encoding posit that sensory systems are optimized to represent sensory information, subject to limitations of noise and resources (e.g., number of neurons, metabolic cost, wiring length). It is difficult to test this concept because optimization of any formulation that attempts to correctly incorporate all of the relevant ingredients is generally intractable. A substantial literature has considered population models in which each neuron's mean response to a scalar variable is characterized by a tuning curve [e.g., 1–6]. For these simplified models, several papers have examined the optimization of Fisher information, as a bound on mean squared error [7–10]. In these results, the distribution of sensory variables is assumed to be uniform and the populations are assumed to be homogeneous with regard to tuning curve shape, spacing, and amplitude.

The distribution of sensory variables encountered in the environment is often non-uniform, and it is thus of interest to understand how variations in probability affect the design of optimal populations. It would seem natural that a neural system should devote more resources to regions of sensory space that occur with higher probability, analogous to results in coding theory [11]. At the single neuron level, several publications describe solutions in which monotonic neural response functions allocate greater dynamic range to higher probability stimuli [12–15]. At the population level, non-uniform allocations of neurons with identical tuning curves have been shown to be optimal for non-uniform stimulus distributions [16, 17].

Here, we examine the influence of a sensory prior on the optimal allocation of neurons and spikes in a population, and the implications of this optimal allocation for subsequent perception. Given a prior distribution over a scalar stimulus parameter, and a resource budget of $N$ neurons with an average of $R$ spikes/sec for the entire population, we seek the optimal shapes, positions, and amplitudes of tuning curves. We assume a population with independent Poisson spiking, and consider a family of objective functions based on Fisher information. We then approximate the Fisher information in terms of two continuous resource variables, the density and gain of the tuning curves. This approximation allows us to obtain a closed form solution for the optimal population. For all objective functions, we find that the optimal tuning curve properties (cell density, tuning width, and gain) are power-law functions of the stimulus prior, with exponents dependent on the specific choice of objective function. Through the Fisher information, we also derive a bound on perceptual discriminability, again in the form a power-law of the stimulus prior. Thus, our framework provides direct and experimentally testable links between sensory priors, properties of the neural representation, and perceptual discriminability. We provide preliminary evidence that these relationships are supported by experimental data.

## 2   Encoding model

We assume a conventional model for a population of $N$ neurons responding to a single scalar variable, $s$ [1–6]. The number of spikes emitted (per unit time) by the $n$th neuron is a sample from an independent Poisson process, with mean rate determined by its tuning function, $h_n(s)$. The probability density of the population response can be written as

$$p(\vec{r}|s) = \prod_{n=1}^{N} \frac{h_n(s)^{r_n}\ e^{-h_n(s)}}{r_n!}.$$

We also assume the total expected spike rate, $R$, of the population is fixed, which places a constraint on the tuning curves:

$$\int p(s) \sum_{n=1}^{N} h_n(s)\ \mathrm{d}s = R, \tag{1}$$

where $p(s)$ is the probability distribution of stimuli in the environment. We refer to this as a sensory prior, in anticipation of its future use in Bayesian decoding of the population response.

## 3   Objective function

We now ask: what is the best way to represent values drawn from $p(s)$ given the limited resources of $N$ neurons and $R$ total spikes? To formulate a family of objective functions which depend on both $p(s)$, and the tuning curves, we first rely on Fisher information, $I_f(s)$, which can be written as a function of the tuning curves [1, 18]:

$$I_f(s) = -\int p(\vec{r}|s)\ \frac{\partial^2}{\partial s^2} \log p(\vec{r}|s)\ \mathrm{d}\vec{r}$$
$$= \sum_{n=1}^{N} \frac{h_n'^2(s)}{h_n(s)}.$$

The Fisher information can be used to express lower bounds on mutual information [16], the variance of an unbiased estimator [18], and perceptual discriminability [19]. Specifically, the mutual information, $I(\vec{r}; s)$, is bounded by:

$$I(\vec{r}; s) \geq H(s) - \frac{1}{2} \int p(s)\ \log\left(\frac{2\pi e}{I_f(s)}\right)\ \mathrm{d}s, \tag{2}$$

where $H(s)$ is the entropy, or amount of information inherent in $p(s)$, which is independent of the neural population. The Cramer-Rao inequality allows us to express the minimum expected squared

stimulus discriminability achievable by any decoder[1]:

$$\delta^2 \geq \Delta^2 \int \frac{p(s)}{I_f(s)} \, \mathrm{d}s. \tag{3}$$

The constant $\Delta$ determines the performance level at threshold in a discrimination task.

We formulate a generalized objective function that includes the Fisher bounds on information and discriminability as special cases:

$$\underset{h_n(s)}{\arg\max} \int p(s) \, f\left(\sum_{n=1}^{N} \frac{h_n'^2(s)}{h_n(s)}\right) \, \mathrm{d}s, \qquad \text{s.t.} \quad \int p(s) \sum_{n=1}^{N} h_n(s) \, \mathrm{d}s = R, \tag{4}$$

where $f(\cdot)$ is either the natural logarithm, or a power function. When $f(x) = \log(x)$, optimizing Eq. (4) is equivalent to maximizing the lower bound on mutual information given in Eq. (2). We refer to this as the *infomax* objective function. Otherwise, we assume $f(x) = x^\alpha$, for some exponent $\alpha$. Optimizing Eq. (4) with $\alpha = -1$ is equivalent to minimizing the squared discriminability bound expressed in Eq. (3). We refer to this as the *discrimax* objective function.

## 4  How to optimize?

The objective function expressed in Eq. (4) is difficult to optimize because it is non-convex. To make the problem tractable, we first introduce a parametrization of the population in terms of cell density and gain. The cell density controls both the spacing *and* width of the tuning curves, and the gain controls their maximum average firing rates. Second, we show that Fisher information can be closely approximated as a continuous function of density and gain. Finally, re-writing the objective function and constraints in these terms allows us to obtain closed-form solutions for the optimal tuning curves.

### 4.1  Density and gain for a homogeneous population

If $p(s)$ is uniform, then by symmetry, the Fisher information for an optimal neural population should also be uniform. We assume a convolutional population of tuning curves, evenly spaced on the unit lattice, such that they approximately "tile" the space:

$$\sum_{n=1}^{N} h(s-n) \approx 1.$$

We also assume that this population has an approximately constant Fisher information:

$$I_f(s) = \sum_{n=1}^{N} \frac{h'^2(s-n)}{h(s-n)}$$

$$= \sum_{n=1}^{N} \phi(s-n) \approx I_{\text{conv}}. \tag{5}$$

That is, we assume that the Fisher information curves for the individual neurons, $\phi(s-n)$, also tile the stimulus space. The value of the constant, $I_{\text{conv}}$, is dependent on the details of the tuning curve shape, $h(s)$, which we leave unspecified. As an example, Fig. 1(a-b) shows that the Fisher information for a convolutional population of Gaussian tuning curves, with appropriate width, is approximately constant.

Now we introduce two scalar values, a gain ($g$), and a density ($d$), that affect the convolutional population as follows:

$$h_n(s) = g \, h\left(d(s - \frac{n}{d})\right). \tag{6}$$

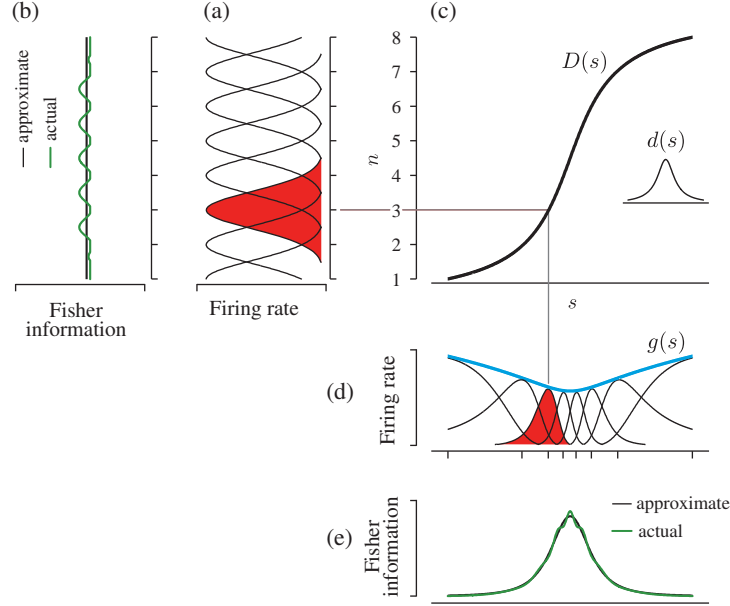

**Fig. 1.** Construction of a heterogeneous population of neurons. **(a)** Homogeneous population with Gaussian tuning curves on the unit lattice. The tuning width of $\sigma = 0.55$ is chosen so that the curves approximately tile the stimulus space. **(b)** The Fisher information of the convolutional population (green) is approximately constant. **(c)** Inset shows $d(s)$, the tuning curve density. The cumulative integral of this density, $D(s)$, alters the positions and widths of the tuning curves in the convolutional population. **(d)** The warped population, with tuning curve peaks (aligned with tick marks, at locations $s_n = D^{-1}(n)$), is scaled by the gain function, $g(s)$ (blue). A single tuning curve is highlighted (red) to illustrate the effect of the warping and scaling operations. **(e)** The Fisher information of the inhomogeneous population is approximately proportional to $d^2(s)g(s)$.

The gain modulates the maximum average firing rate of each neuron in the population. The density controls both the spacing and width of the tuning curves: as the density increases, the tuning curves become narrower, and are spaced closer together so as to maintain their tiling of stimulus space. The effect of these two parameters on Fisher information is:

$$I_f(s) = d^2 g \sum_{n=1}^{N(d)} \phi(ds - n)$$

$$\approx d^2 g \, I_{\text{conv}}.$$

The second line follows from the assumption of Eq. (5), that the Fisher information of the convolutional population is approximately constant with respect to $s$.

The total resources, $N$ and $R$, naturally constrain $d$ and $g$, respectively. If the original (unit-spacing) convolutional population is supported on the interval $(0, Q)$ of the stimulus space, then the number of neurons in the modulated population must be $N(d) = Qd$ to cover the same interval. Under the assumption that the tuning curves tile the stimulus space, Eq. (1) implies that $R = g$ for the modulated population.

## 4.2 Density and gain for a heterogeneous population

Intuitively, if $p(s)$ is non-uniform, the optimal Fisher information should also be non-uniform. This can be achieved through inhomogeneities in either the tuning curve density or gain. We thus generalize density and gain to be continuous functions of the stimulus, $d(s)$ and $g(s)$, that warp and scale the convolutional population:

$$h_n(s) = g(s_n) \, h(D(s) - n). \tag{7}$$

| | | Infomax | Discrimax | General |
|---|---|---|---|---|
| Optimized function: | | $f(x) = \log x$ | $f(x) = -x^{-1}$ | $f(x) = -x^{\alpha}, \alpha < \frac{1}{3}$ |
| **Density (Tuning width)**$^{-1}$ | $d(s)$ | $Np(s)$ | $Np^{\frac{1}{2}}(s)$ | $Np^{\frac{\alpha-1}{3\alpha-1}}(s)$ |
| **Gain** | $g(s)$ | $R$ | $Rp^{-\frac{1}{2}}(s)$ | $Rp^{\frac{2\alpha}{1-3\alpha}}(s)$ |
| **Fisher information** | $I_f(s)$ | $\propto RN^2 p^2(s)$ | $\propto RN^2 p^{\frac{1}{2}}(s)$ | $\propto RN^2 p^{\frac{2}{1-3\alpha}}(s)$ |
| **Discriminability bound** | $\delta_{\min}(s)$ | $\propto p^{-1}(s)$ | $\propto p^{-\frac{1}{4}}(s)$ | $\propto p^{\frac{1}{3\alpha-1}}(s)$ |

**Table 1.** Optimal heterogeneous population properties, for objective functions specified by Eq. (9).

Here, $D(s) = \int_{-\infty}^{s} d(t)dt$, the cumulative integral of $d(s)$, warps the shape of the prototype tuning curve. The value $s_n = D^{-1}(n)$ represents the preferred stimulus value of the (warped) $n$th tuning curve (Fig. 1(b-d)). Note that the warped population retains the tiling properties of the original convolutional population. As in the uniform case, the density controls both the spacing and width of the tuning curves. This can be seen by rewriting Eq. (7) as a first-order Taylor expansion of $D(s)$ around $s_n$:

$$h_n(s) \approx g(s_n)\, h(d(s_n)(s - s_n)),$$

which is a generalization of Eq. (6).

We can now write the Fisher information of the heterogeneous population of neurons in Eq. (7) as

$$I_f(s) = \sum_{n=1}^{N} d^2(s)\, g(s_n)\, \phi(D(s) - n)$$
$$\approx d^2(s)\, g(s)\, I_{\text{conv}}. \tag{8}$$

In addition to assuming that the Fisher information is approximately constant (Eq. (5)), we have also assumed that $g(s)$ is smooth relative to the width of $\phi(D(s) - n)$ for all $n$, so that we can approximate $g(s_n)$ as $g(s)$ and remove it from the sum. The end result is an approximation of Fisher information in terms of the continuous parameterization of cell density and gain. As earlier, the constant $I_{\text{conv}}$ is determined by the precise shape of the tuning curves.

As in the homogeneous case, the global resource values $N$ and $R$ will place constraints on $d(s)$ and $g(s)$, respectively. In particular, we require that $D(\cdot)$ map the entire input space onto the range $[1, N]$, and thus $D(\infty) = N$, or equivalently, $\int d(s)\,\mathrm{d}s = N$. To attain the proper rate, we use the fact that the warped tuning curves sum to unity (before multiplication by the gain function) and use Eq. (1) to obtain the constraint $\int p(s)g(s)\,\mathrm{d}s = R$.

### 4.3 Objective function and solution for a heterogeneous population

Approximating Fisher information as proportional to squared density and gain allows us to re-write the objective function and resource constraints of Eq. (4) as

$$\operatorname*{arg\,max}_{d(s),g(s)} \int p(s)\, f\left(d^2(s)\, g(s)\right)\,\mathrm{d}s, \quad \text{s.t.} \quad \int d(s)\,\mathrm{d}s = N, \quad \text{and} \quad \int p(s)g(s)\,\mathrm{d}s = R. \tag{9}$$

A closed-form optimum of this objective function is easily determined by taking the gradient of the Lagrangian, setting to zero, and solving the resulting system of equations. Solutions are provided in Table 1 for the infomax, discrimax, and general power cases.

In all cases, the solution specifies a power-law relationship between the prior, and the density and gain of the tuning curves. In general, all solutions allocate more neurons, with correspondingly narrower tuning curves, to higher-probability stimuli. In particular, the infomax solution allocates an approximately equal amount of probability mass to each neuron. The shape of the optimal gain function depends on the objective function: for $\alpha < 0$, neurons with lower firing rates are used to represent stimuli with higher probabilities, and for $\alpha > 0$, neurons with higher firing rates are used for stimuli with higher probabilities. Note also that the global resource values, $N$ and $R$, enter only as scale factors on the overall solution, allowing us to easily test the validity of the

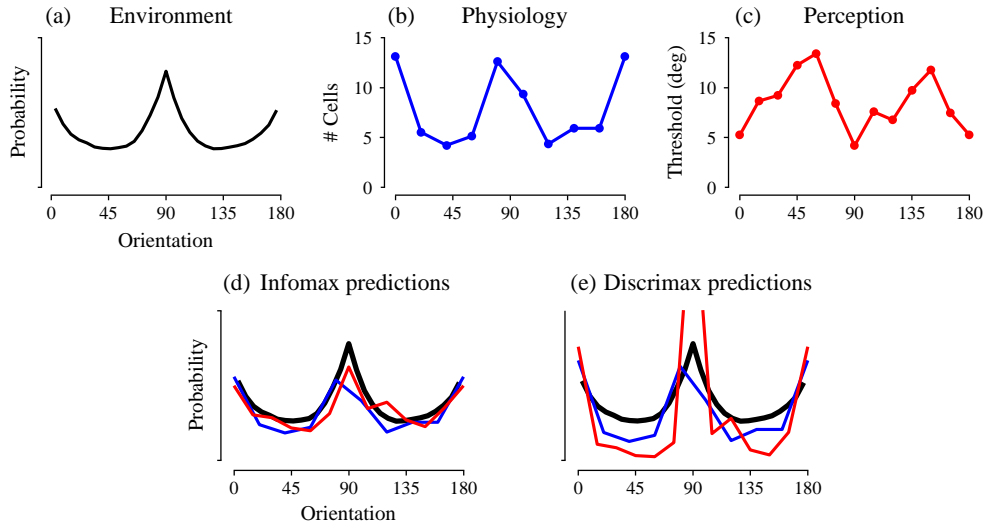

**Fig. 2. (a)** Distribution of orientations averaged across three natural image databases [20–22]. **(b)** Density, or total number of Macaque V1 cells tuned to each preferred orientation [23]. **(c)** Orientation discrimination thresholds averaged across four human subjects [24]. **(d & e)** Infomax and discrimax predictions of orientation distribution. Blue: prediction from cell density. Red: prediction from discrimination thresholds. Predictions were made by exponentiating the raw data with the appropriate exponent from Table 1, then normalizing to integrate to one.

predicted relationships on experimental data. In addition to power-law relationships between tuning properties and sensory priors, our formulation offers a direct relationship between the sensory prior and perceptual discriminability. This can be obtained by substituting the optimal solutions for $d(s)$ and $g(s)$ into Eq. (8), and using the resulting Fisher information to bound the discriminability, $\delta(s) \geq \delta_{\min}(s) \equiv \Delta / \sqrt{I_f(s)}$ [19]. The resulting expressions are provided in Table 1.

## 5 Experimental evidence

Our framework predicts a quantitative link between the sensory prior, physiological parameters (the density, tuning widths, and gain of cells), and psychophysically measured discrimination thresholds. We obtained subsets of these quantities for two visual stimulus variables, orientation and spatial frequency, both of believed to be encoded by cells in primary visual cortex (area V1). For each variable, we use the infomax and discrimax solutions to convert the physiological and perceptual measurements, using the appropriate exponents from Table 1, into predictions of the stimulus prior $\hat{p}(s)$. We then compare these predictions to the empirically measured prior $p(s)$.

### 5.1 Orientation

We estimated the prior distribution of orientations in the environment by averaging orientation statistics across three natural image databases. Two databases consist entirely of natural scenes [20, 21], and the third contains natural and manmade scenes [22]. Orientation statistics depend on scale, so we measured statistics at a scale matching the psychophysical experiment from which we obtained perceptual data. The average distribution of orientations exhibits higher probability at the cardinal orientations (vertical and horizontal) than at the oblique orientations (Fig. 2(a)). Measurements of cell density for a population of 79 orientation-tuned V1 cells in Macaque [23] show more cells tuned to the cardinal orientations than the oblique orientations (Fig. 2(b)). Finally, perceptual discrimination thresholds, averaged across four human subjects [24] show a similar bias (Fig. 2(c)), with humans better able to discriminate orientations near the cardinal directions.

All of the orientation data exhibit similar biases, but our theory makes precise and testable predictions about these relationships. If a neural population is designed to maximize information, then the cell density and inverse discrimination thresholds should match the stimulus prior, as expressed in infomax column of Table 1. We normalize these predictions to integrate to one (since the theory

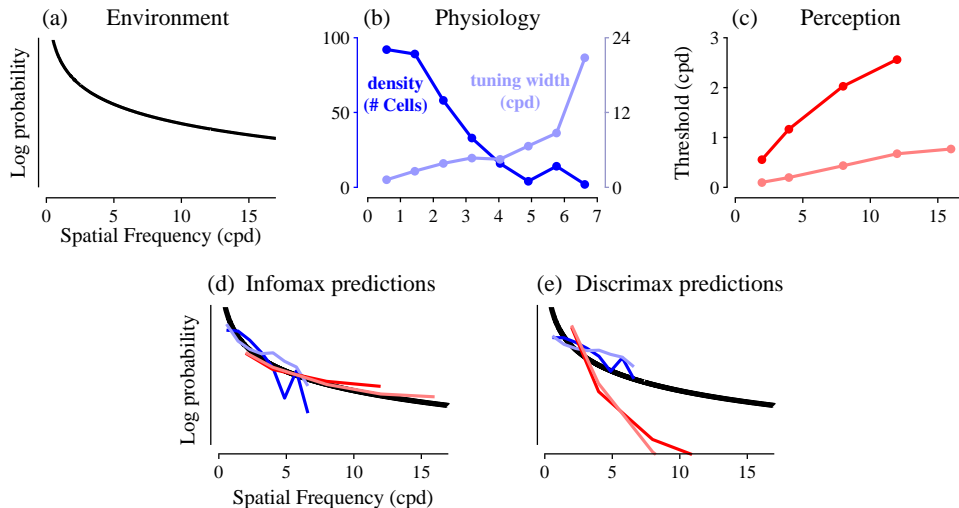

**Fig. 3.** **(a)** Distribution of spatial frequencies computed across two natural image databases [20, 21]. **(b)** Cell density as a function of preferred spatial frequency for a population of 317 V1 cells [25, 28] Dark blue: average number of cells tuned to each spatial frequency. Light blue: average tuning width. **(c)** Average spatial frequency discrimination thresholds. Dark red: thresholds obtained at 10% contrast averaged across 3 human subjects [26]. Light red: thresholds obtained at 25% contrast averaged across 7-13 human subjects [27]. **(d & e)** Infomax and discrimax predictions of spatial frequency distribution. Blues: predictions from cell density and tuning widths. Reds: predictions from discrimination thresholds.

provides only the shapes of the functions, up to unknown values of the resource variables $N$ and $R$), and plot them against the measured prior (Fig. 2(d)). We see that the predictions arising from cell density and discrimination thresholds are consistent with one another, and both are consistent with the stimulus prior. This is especially remarkable given that the measurements come from very different domains (in the case of the perceptual and physiological data, different species). For the discrimax objective function, the exponents in the power-law relationships (expressed in Table 1) are too small, resulting in poor qualitative agreement between the stimulus prior and predictions from the physiology and perception (Fig. 2(e)). For example, predicting the prior from perceptual data, under the discrimax objective function, requires exponentiating discrimination thresholds to the fourth power, resulting in an over exaggeration of the cardinal bias.

## 5.2 Spatial frequency

We obtained a prior distribution over spatial frequencies averaged across two natural image databases [20, 21]. For each image, we computed the magnitude spectrum, and averaged over orientation. We averaged these across images, and fit the result with a power law of exponent $-1.3$ (Fig. 3(a)). We also obtained spatial frequency tuning properties for a population of 317 V1 cells [25]. On average, we see there are more cells, with correspondingly narrower tuning widths, tuned to low spatial frequencies (Fig. 3(b)). These data support the model assumption that tuning width is inversely proportional to cell density. We also obtained average discrimination thresholds for sinusoidal gratings of different spatial frequencies from two studies (Fig. 3(c)). The gratings were shown at 10% contrast to 3 human subjects for one study [26], and 25% contrast for $7 - 13$ human subjects for the other [27]. The thresholds show that, on average, humans are better at discriminating low spatial frequencies.

We again test the infomax and discrimax solutions by comparing predicted distributions obtained from the physiological and perceptual data, to the measured prior. We normalize each prediction to integrate to the corresponding area under the prior. The infomax case shows striking agreement between the measured stimulus prior, and predictions based on the physiological and perceptual measurements (Fig. 3(d)). However, as in the orientation case, discrimax predictions are poor (Fig. 3(e)), suggesting that information maximization provides a better optimality principle for explaining the neural and perceptual encoding of spatial frequency than discrimination maximization.

# 6 Discussion

We have examined the influence sensory priors on the optimal allocation of neural resources, as well as the influence of these optimized resources on subsequent perception. For a family of objective functions, we obtain closed-form solutions specifying power law relationships between the probability distribution of a sensory variable encountered in the environment, the tuning properties of a population that encodes that variable, and the minimum perceptual discrimination thresholds achievable for that variable. We've shown preliminary supportive evidence for these relationships for two different perceptual attributes.

Our analysis requires several approximations and assumptions in order to arrive at an analytical solution. We first rely on lower bounds on mutual information and discriminability based on Fisher information. Fisher information is known to provide a poor bound on mutual information when there are a small number of neurons, a short decoding time, or non-smooth tuning curves [16, 29]. It also provides a poor bound on supra-threshold discriminability [30, 31]. But note that we do not require the bounds on either information or discriminability to be tight, but rather that their optima be close to that of their corresponding true objective functions. We also made several assumptions in deriving our results: (1) the tuning curves, $h(D(s)-n)$, evenly tile the stimulus space; (2) the single neuron Fisher informations, $\phi(D(s) - n)$, evenly tile the stimulus space; and (3) the gain function, $g(s)$, varies slowly and smoothly over the width of $\phi(D(s) - n)$. These assumptions allow us to approximate Fisher information in terms of cell density and gain (Fig. 1(e)), to express the resource constraints in simple form, and to obtain a closed-form solution to the optimization problem.

Our framework offers an important generalization of the population coding literature, allowing for non-uniformity of sensory priors, and corresponding heterogeneity in tuning and gain properties. Nevertheless, it suffers from many of the same simplifications found in previous literature. First, neural spike trains are not Poisson, and they are (at least in some cases) correlated [32]. Second, tuning curve encoding models only specify neural responses to single stimulus values. The model should be generalized to handle arbitrary combinations of stimuli. And third, the response model should be generalized to handle multi-dimensional sensory inputs. Each of these limitations offers an important opportunity for future work.

Finally, our encoding model has direct implications for Bayesian decoding, a problem that has received much attention in recent literature [e.g., 5, 6, 33–35]. A Bayesian decoder must have knowledge of prior probabilities, but it is unclear how such knowledge is obtained or represented in the brain [34]. Previous studies assume that prior probabilities are either uniform [6], represented in the spiking activity of a separate population of neurons [5], or represented (in sample form) in the spontaneous activity [35]. Our encoding formulation provides a mechanism whereby the prior is implicitly encoded in the density and gains of tuning curves, which presumably arise from the strength of synaptic connections. We are currently exploring the requirements for a decoder that can correctly utilize this form of embedded prior information to obtain Bayesian estimates of stimulus variables.

## Footnotes

[1]The conventional Cramer-Rao bound expresses the minimum mean squared error of any estimator, and in general requires a correction for the estimator bias [18]. Here, we use it to bound the squared *discriminability* of the estimator, as expressed in the stimulus space, which is independent of bias [19].

## References

[1] HS Seung and H Sompolinsky. Simple models for reading neuronal population codes. *Proc. Natl. Acad. Sci. U.S.A.*, 90:10749–10753, Nov 1993.

[2] RS Zemel, P Dayan, and A Pouget. Probabilistic interpretation of population codes. *Neural Comput*, 10(2):403–430, Feb 1998.

[3] A Pouget, P Dayan, and RS Zemel. Inference and computation with population codes. *Annu Rev Neurosci*, 26:381–410, 2003.

[4] TD Sanger. Neural population codes. *Curr Opin Neurobiol*, 13(2):238–249, Apr 2003.

[5] WJ Ma, JM Beck, PE Latham, and A Pouget. Bayesian inference with probabilistic population codes. *Nat Neurosci*, 9(11):1432–1438, Nov 2006.

[6] M Jazayeri and JA Movshon. Optimal representation of sensory information by neural populations. *Nat. Neurosci.*, 9:690–696, May 2006.

[7] K Zhang and TJ Sejnowski. Neuronal tuning: To sharpen or broaden? *Neural Comput*, 11(1):75–84, Jan 1999.

[8] A Pouget, S Deneve, JC Ducom, and PE Latham. Narrow versus wide tuning curves: What's best for a population code? *Neural Comput*, 11(1):85–90, Jan 1999.

[9] WM Brown and A Bäcker. Optimal neuronal tuning for finite stimulus spaces. *Neural Comput*, 18(7):1511–1526, Jul 2006.

[10] MA Montemurro and S Panzeri. Optimal tuning widths in population coding of periodic variables. *Neural Comput*, 18(7):1555–1576, Jul 2006.

[11] A Gersho and RM Gray. *Vector quantization and signal compression*. Kluwer Academic Publishers, Norwell, MA, 1991.

[12] S Laughlin. A simple coding procedure enhances a neuron's information capacity. *Z. Naturforschung*, 36c:910–912, 1981.

[13] JP Nadal and N Parga. Nonlinear neurons in the low-noise limit: a factorial code maximizes information transfer. *Network: Computation in Neural Systems*, 5:565–581(17), 1994.

[14] T von der Twer and DI MacLeod. Optimal nonlinear codes for the perception of natural colours. *Network*, 12(3):395–407, Aug 2001.

[15] MD McDonnell and NG Stocks. Maximally informative stimuli and tuning curves for sigmoidal rate-coding neurons and populations. *Phys Rev Lett*, 101:58103–58107, 2008.

[16] N Brunel and JP Nadal. Mutual information, Fisher information, and population coding. *Neural Comput*, 10(7):1731–1757, Oct 1998.

[17] NS Harper and D McAlpine. Optimal neural population coding of an auditory spatial cue. *Nature*, 430(7000):682–686, Aug 2004.

[18] D Cox and D Hinkley. *Theoretical statistics*. London: Chapman and Hall., 1974.

[19] P Seriés, AA Stocker, and EP Simoncelli. Is the homunculus "aware" of sensory adaptation? *Neural Comput*, 21(12):3271–3304, Dec 2009.

[20] JH van Hateren and A van der Schaaf. Independent component filters of natural images compared with simple cells in primary visual cortex. *Proc Biol Sci*, 265(1394):359–366, Mar 1998.

[21] E Doi, T Inui, TW Lee, T Wachtler, and TJ Sejnowski. Spatiochromatic receptive field properties derived from information-theoretic analyses of cone mosaic responses to natural scenes. *Neural Comput*, 15(2):397–417, Feb 2003.

[22] A Olmos and FAA Kingdom. McGill calibrated image database, http://tabby.vision.mcgill.ca, 2004.

[23] RJ Mansfield. Neural basis of orientation perception in primate vision. *Science*, 186(4169):1133–1135, Dec 1974.

[24] AR Girshick, MS Landy, and EP Simoncelli. Bayesian line orientation perception: Human prior expectations match natural image statistics. In *Frontiers in Systems Neuroscience (CoSyNe).*, 2010.

[25] JR Cavanaugh, W Bair, and JA Movshon. Selectivity and spatial distribution of signals from the receptive field surround in macaque v1 neurons. *J Neurophysiol*, 88(5):2547–2556, Nov 2002.

[26] T Caelli, H Brettel, I Rentschler, and R Hilz. Discrimination thresholds in the two-dimensional spatial frequency domain. *Vision Res*, 23(2):129–133, 1983.

[27] D Regan, S Bartol, TJ Murray, and KI Beverley. Spatial frequency discrimination in normal vision and in patients with multiple sclerosis. *Brain*, 105 (Pt 4):735–754, Dec 1982.

[28] JR Cavanaugh, W Bair, and JA Movshon. Nature and interaction of signals from the receptive field center and surround in macaque v1 neurons. *J Neurophysiol*, 88(5):2530–2546, Nov 2002.

[29] M Bethge, D Rotermund, and K Pawelzik. Optimal short-term population coding: when fisher information fails. *Neural Comput*, 14(10):2317–2351, Oct 2002.

[30] M Shamir and H Sompolinsky. Implications of neuronal diversity on population coding. *Neural Comput*, 18(8):1951–1986, Aug 2006.

[31] P Berens, S Gerwinn, A Ecker, and M Bethge. Neurometric function analysis of population codes. In *Advances in Neural Information Processing Systems 22*, pages 90–98, 2009.

[32] E Zohary, MN Shadlen, and WT Newsome. Correlated neuronal discharge rate and its implications for psychophysical performance. *Nature*, 370(6485):140–143, Jul 1994.

[33] DC Knill and A Pouget. The bayesian brain: the role of uncertainty in neural coding and computation. *Trends Neurosci*, 27(12):712–719, Dec 2004.

[34] EP Simoncelli. Optimal estimation in sensory systems. In M Gazzaniga, editor, *The Cognitive Neurosciences, IV*, chapter 36, pages 525–535. MIT Press, Oct 2009.

[35] J Fiser, P Berkes, G Orbán, and M Lengyel. Statistically optimal perception and learning: from behavior to neural representations. *Trends Cogn Sci*, 14(3):119–130, Mar 2010.

